# Multiplicative Updates for Nonnegative Quadratic Programming in Support Vector Machines

**Fei Sha**[1], **Lawrence K. Saul**[1], and **Daniel D. Lee**[2]
[1]Department of Computer and Information Science
[2]Department of Electrical and System Engineering
University of Pennsylvania
200 South 33rd Street, Philadelphia, PA 19104
{*feisha,lsaul*}*@cis.upenn.edu, ddlee@ee.upenn.edu*

## Abstract

We derive multiplicative updates for solving the nonnegative quadratic programming problem in support vector machines (SVMs). The updates have a simple closed form, and we prove that they converge monotonically to the solution of the maximum margin hyperplane. The updates optimize the traditionally proposed objective function for SVMs. They do not involve any heuristics such as choosing a learning rate or deciding which variables to update at each iteration. They can be used to adjust all the quadratic programming variables in parallel with a guarantee of improvement at each iteration. We analyze the asymptotic convergence of the updates and show that the coefficients of non-support vectors decay geometrically to zero at a rate that depends on their margins. In practice, the updates converge very rapidly to good classifiers.

## 1   Introduction

Support vector machines (SVMs) currently provide state-of-the-art solutions to many problems in machine learning and statistical pattern recognition[18]. Their superior performance is owed to the particular way they manage the tradeoff between bias (underfitting) and variance (overfitting). In SVMs, kernel methods are used to map inputs into a higher, potentially infinite, dimensional feature space; the decision boundary between classes is then identified as the maximum margin hyperplane in the feature space. While SVMs provide the flexibility to implement highly nonlinear classifiers, the maximum margin criterion helps to control the capacity for overfitting. In practice, SVMs generalize very well — even better than their theory suggests.

Computing the maximum margin hyperplane in SVMs gives rise to a problem in nonnegative quadratic programming. The resulting optimization is convex, but due to the nonnegativity constraints, it cannot be solved in closed form, and iterative solutions are required. There is a large literature on iterative algorithms for nonnegative quadratic programming in general and for SVMs as a special case[3, 17]. Gradient-based methods are the simplest possible approach, but their convergence depends on careful selection of the learning rate, as well as constant attention to the nonnegativity constraints which may not be naturally enforced. Multiplicative updates based on exponentiated gradients (EG)[5, 10] have been

investigated as an alternative to traditional gradient-based methods. Multiplicative updates are naturally suited to sparse nonnegative optimizations, but EG updates—like their additive counterparts—suffer the drawback of having to choose a learning rate.

Subset selection methods constitute another approach to the problem of nonnegative quadratic programming in SVMs. Generally speaking, these methods split the variables at each iteration into two sets: a *fixed* set in which the variables are held constant, and a *working* set in which the variables are optimized by an internal subroutine. At the end of each iteration, a heuristic is used to transfer variables between the two sets and improve the objective function. An extreme version of this approach is the method of Sequential Minimal Optimization (SMO)[15], which updates only two variables per iteration. In this case, there exists an analytical solution for the updates, so that one avoids the expense of a potentially iterative optimization within each iteration of the main loop.

In general, despite the many proposed approaches for training SVMs, solving the quadratic programming problem remains a bottleneck in their implementation. (Some researchers have even advocated changing the objective function in SVMs to simplify the required optimization[8, 13].) In this paper, we propose a new iterative algorithm, called **Multiplicative Margin Maximization** ($M^3$), for training SVMs. The $M^3$ updates have a simple closed form and converge monotonically to the solution of the maximum margin hyperplane. They do not involve heuristics such as the setting of a learning rate or the switching between fixed and working subsets; all the variables are updated in parallel. They provide an extremely straightforward way to implement traditional SVMs. Experimental and theoretical results confirm the promise of our approach.

## 2   Nonnegative quadratic programming

We begin by studying the general problem of nonnegative quadratic programming. Consider the minimization of the quadratic objective function

$$F(\mathbf{v}) = \frac{1}{2}\mathbf{v}^T\mathbf{A}\mathbf{v} + \mathbf{b}^T\mathbf{v}, \tag{1}$$

subject to the constraints that $v_i \geq 0 \; \forall i$. We assume that the matrix $\mathbf{A}$ is symmetric and semipositive definite, so that the objective function $F(\mathbf{v})$ is bounded below, and its optimization is convex. Due to the nonnegativity constraints, however, there does not exist an analytical solution for the global minimum (or minima), and an iterative solution is needed.

### 2.1   Multiplicative updates

Our iterative solution is expressed in terms of the positive and negative components of the matrix $\mathbf{A}$ in eq. (1). In particular, let $\mathbf{A}^+$ and $\mathbf{A}^-$ denote the *nonnegative* matrices:

$$A_{ij}^+ = \begin{cases} A_{ij} & \text{if } A_{ij} > 0, \\ 0 & \text{otherwise,} \end{cases} \quad \text{and} \quad A_{ij}^- = \begin{cases} |A_{ij}| & \text{if } A_{ij} < 0, \\ 0 & \text{otherwise.} \end{cases} \tag{2}$$

It follows trivially that $\mathbf{A} = \mathbf{A}^+ - \mathbf{A}^-$. In terms of these nonnegative matrices, our proposed updates (to be applied in parallel to all the elements of $\mathbf{v}$) take the form:

$$v_i \longleftarrow v_i \left[ \frac{-b_i + \sqrt{b_i^2 + 4(\mathbf{A}^+\mathbf{v})_i(\mathbf{A}^-\mathbf{v})_i}}{2(\mathbf{A}^+\mathbf{v})_i} \right]. \tag{3}$$

The iterative updates in eq. (3) are remarkably simple to implement. Their somewhat mysterious form will be clarified as we proceed. Let us begin with two simple observations. First, eq. (3) prescribes a multiplicative update for the $i^{\text{th}}$ element of $\mathbf{v}$ in terms of the $i^{\text{th}}$ elements of the vectors $\mathbf{b}$, $\mathbf{A}^+\mathbf{v}$, and $\mathbf{A}^+\mathbf{v}$. Second, since the elements of $\mathbf{v}$, $\mathbf{A}^+$, and $\mathbf{A}^-$ are nonnegative, the overall factor multiplying $v_i$ on the right hand side of eq. (3) is always nonnegative. Hence, these updates never violate the constraints of nonnegativity.

## 2.2 Fixed points

We can show further that *these updates have fixed points wherever the objective function, $F(\mathbf{v})$ achieves its minimum value*. Let $\mathbf{v}^*$ denote a global minimum of $F(\mathbf{v})$. At such a point, one of two conditions must hold for each element $v_i^*$: either (i) $v_i^* > 0$ and $(\partial F/\partial v_i)|_{\mathbf{v}^*} = 0$, or (ii), $v_i^* = 0$ and $(\partial F/\partial v_i)|_{\mathbf{v}^*} \geq 0$. The first condition applies to the positive elements of $\mathbf{v}^*$, whose corresponding terms in the gradient must vanish. These derivatives are given by:

$$\left.\frac{\partial F}{\partial v_i}\right|_{\mathbf{v}^*} = (\mathbf{A}^+ \mathbf{v}^*)_i - (\mathbf{A}^- \mathbf{v}^*)_i + b_i. \tag{4}$$

The second condition applies to the zero elements of $\mathbf{v}^*$. Here, the corresponding terms of the gradient must be nonnegative, thus pinning $v_i^*$ to the boundary of the feasibility region.

The multiplicative updates in eq. (3) have fixed points wherever the conditions for global minima are satisfied. To see this, let

$$\gamma_i \overset{\triangle}{=} \frac{-b_i + \sqrt{b_i^2 + 4(\mathbf{A}^+ \mathbf{v}^*)_i (\mathbf{A}^- \mathbf{v}^*)_i}}{2(\mathbf{A}^+ \mathbf{v}^*)_i} \tag{5}$$

denote the factor multiplying the $i^{\text{th}}$ element of $\mathbf{v}$ in eq. (3), evaluated at $\mathbf{v}^*$. Fixed points of the multiplicative updates occur when one of two conditions holds for each element $v_i$: either (i) $v_i^* > 0$ and $\gamma_i = 1$, or (ii) $v_i^* = 0$. It is straightforward to show from eqs. (4–5) that $(\partial F/\partial v_i)|_{\mathbf{v}^*} = 0$ implies $\gamma_i = 1$. Thus the conditions for global minima establish the conditions for fixed points of the multiplicative updates.

## 2.3 Monotonic convergence

The updates not only have the correct fixed points; they also lead to monotonic improvement in the objective function, $F(\mathbf{v})$. This is established by the following theorem:

**Theorem 1** *The function $F(\mathbf{v})$ in eq. (1) decreases monotonically to the value of its global minimum under the multiplicative updates in eq. (3).*

The proof of this theorem (sketched in Appendix A) relies on the construction of an auxiliary function which provides an upper bound on $F(\mathbf{v})$. Similar methods have been used to prove the convergence of many algorithms in machine learning[1, 4, 6, 7, 12, 16].

# 3 Support vector machines

We now consider the problem of computing the maximum margin hyperplane in SVMs[3, 17, 18]. Let $\{(\mathbf{x}_i, y_i)\}_{i=1}^{N}$ denote labeled examples with binary class labels $y_i = \pm 1$, and let $K(\mathbf{x}_i, \mathbf{x}_j)$ denote the kernel dot product between inputs. In this paper, we focus on the simple case where in the high dimensional feature space, the classes are linearly separable and the hyperplane is required to pass through the origin[1]. In this case, the maximum margin hyperplane is obtained by minimizing the loss function:

$$L(\alpha) = -\sum_i \alpha_i + \frac{1}{2} \sum_{ij} \alpha_i \alpha_j y_i y_j K(\mathbf{x}_i, \mathbf{x}_j), \tag{6}$$

subject to the nonnegativity constraints $\alpha_i \geq 0$. Let $\alpha^*$ denote the location of the minimum of this loss function. The maximal margin hyperplane has normal vector $\mathbf{w} = \sum_i \alpha_i^* y_i \mathbf{x}_i$ and satisfies the margin constraints $y_i K(\mathbf{w}, \mathbf{x_i}) \geq 1$ for all examples in the training set.

| Kernel | Polynomial | | Radial | | |
|---|---|---|---|---|---|
| Data | $k\!=\!4$ | $k\!=\!6$ | $\sigma\!=\!0.3$ | $\sigma\!=\!1.0$ | $\sigma\!=\!3.0$ |
| Sonar | 9.6% | 9.6% | 7.6% | 6.7% | 10.6% |
| Breast cancer | 5.1% | 3.6% | 4.4% | 4.4% | 4.4% |

Table 1: Misclassification error rates on the sonar and breast cancer data sets after 512 iterations of the multiplicative updates.

### 3.1 Multiplicative updates

The loss function in eq. (6) is a special case of eq. (1) with $A_{ij} = y_i y_j K(\mathbf{x}_i, \mathbf{x}_j)$ and $b_i = -1$. Thus, the multiplicative updates for computing the maximal margin hyperplane in hard margin SVMs are given by:

$$\alpha_i \longleftarrow \alpha_i \left[ \frac{1 + \sqrt{1 + 4(\mathbf{A}^+\alpha)_i(\mathbf{A}^-\alpha)_i}}{2(\mathbf{A}^+\alpha)_i} \right] \qquad (7)$$

where $\mathbf{A}^\pm$ are defined as in eq. (2). We will refer to the learning algorithm for hard margin SVMs based on these updates as Multiplicative Margin Maximization ($M^3$).

It is worth comparing the properties of these updates to those of other approaches. Like multiplicative updates based on exponentiated gradients (EG)[5, 10], the $M^3$ updates are well suited to sparse nonnegative optimizations[2]; unlike EG updates, however, they do not involve a learning rate, and they come with a guarantee of monotonic improvement. Like the updates for Sequential Minimal Optimization (SMO)[15], the $M^3$ updates have a simple closed form; unlike SMO updates, however, they can be used to adjust all the quadratic programming variables in parallel (or any subset thereof), not just two at a time. Finally, we emphasize that the $M^3$ updates optimize the traditional objective function for SVMs; they do not compromise the goal of computing the maximal margin hyperplane.

### 3.2 Experimental results

We tested the effectiveness of the multiplicative updates in eq. (7) on two real world problems: binary classification of aspect-angle dependent sonar signals[9] and breast cancer data[14]. Both data sets, available from the UCI Machine Learning Repository[2], have been widely used to benchmark many learning algorithms, including SVMs[5]. The sonar and breast cancer data sets consist of 208 and 683 labeled examples, respectively. Training and test sets for the breast cancer experiments were created by 80%/20% splits of the available data.

We experimented with both polynomial and radial basis function kernels. The polynomial kernels had degrees $k\!=\!4$ and $k\!=\!6$, while the radial basis function kernels had variances of $\sigma\!=\!0.3, 1.0$ and $3.0$. The coefficients $\alpha_i$ were uniformly initialized to a value of one in all experiments.

Misclassification rates on the test data sets after 512 iterations of the multiplicative updates are shown in Table 1. As expected, the results match previously published error rates on these data sets[5], showing that the $M^3$ updates do in practice converge to the maximum margin hyperplane. Figure 1 shows the rapid convergence of the updates to good classifiers in just one or two iterations.

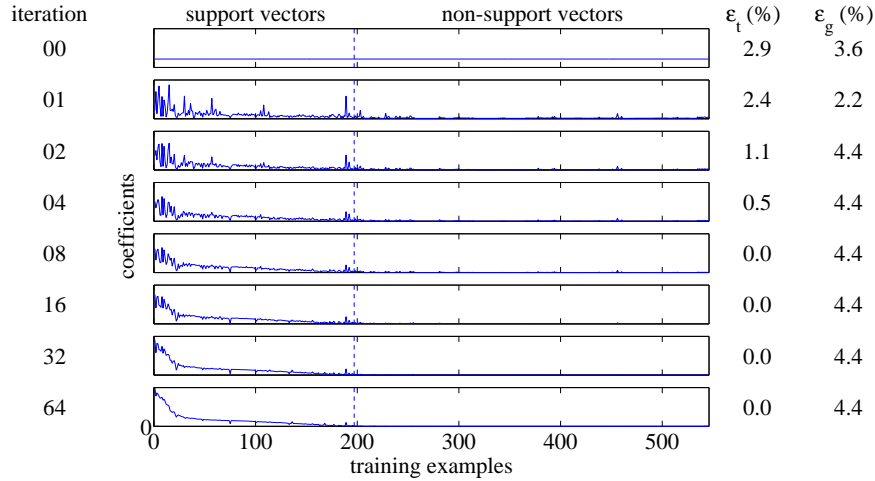

Figure 1: Rapid convergence of the multiplicative updates in eq. (7). The plots show results after different numbers of iterations on the breast cancer data set with the radial basis function kernel ($\sigma = 3$). The horizontal axes index the coefficients $\alpha_i$ of the 546 training examples; the vertical axes show their values. For ease of visualization, the training examples were ordered so that support vectors appear to the left and non-support vectors, to the right. The coefficients $\alpha_i$ were uniformly initialized to a value of one. Note the rapid attenuation of non-support vector coefficients after one or two iterations. Intermediate error rates on the training set ($\epsilon_t$) and test set ($\epsilon_g$) are also shown.

### 3.3   Asymptotic convergence

The rapid decay of non-support vector coefficients in Fig. 1 motivated us to analyze their rates of asymptotic convergence. Suppose we perturb just one of the non-support vector coefficients in eq. (6)—say $\alpha_i$–away from the fixed point to some small nonzero value $\delta\alpha_i$. If we hold all the variables but $\alpha_i$ fixed and apply its multiplicative update, then the new displacement $\delta\alpha'_i$ after the update is given asymptotically by $(\delta\alpha'_i) \approx (\delta\alpha_i)\gamma_i$, where

$$\gamma_i = \frac{1 + \sqrt{1 + 4(\mathbf{A}^+\alpha^*)_i(\mathbf{A}^-\alpha^*)_i}}{2(\mathbf{A}^+\alpha^*)_i}, \tag{8}$$

and $A_{ij} = y_iy_jK(\mathbf{x}_i, \mathbf{x}_j)$. (Eq. (8) is merely the specialization of eq. (5) to SVMs.) We can thus bound the asymptotic rate of convergence—in this idealized but instructive setting— by computing an upper bound on $\gamma_i$, which determines how fast the perturbed coefficient decays to zero. (Smaller $\gamma_i$ implies faster decay.) In general, the asymptotic rate of convergence is determined by the overall positioning of the data points and classification hyperplane in the feature space. The following theorem, however, provides a simple bound in terms of easily understood geometric quantities.

**Theorem 2** *Let $d_i = |K(\mathbf{x}_i, \mathbf{w})|/\sqrt{K(\mathbf{w}, \mathbf{w})}$ denote the perpendicular distance in the feature space from $\mathbf{x}_i$ to the maximum margin hyperplane, and let $d = \min_j d_j = 1/\sqrt{K(\mathbf{w}, \mathbf{w})}$ denote the one-sided margin of the classifier. Also, let $\ell_i = \sqrt{K(\mathbf{x}_i, \mathbf{x}_i)}$ denote the distance of $\mathbf{x}_i$ to the origin in the feature space, and let $\ell = \max_j \ell_j$ denote the largest such distance. Then a bound on the asymptotic rate of convergence $\gamma_i$ is given by:*

$$\gamma_i \leq \left[1 + \frac{1}{2}\frac{(d_i - d)d}{\ell_i\ell}\right]^{-1}. \tag{9}$$

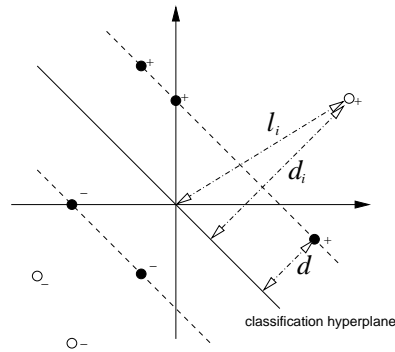

Figure 2: Quantities used to bound the asymptotic rate of convergence in eq. (9); see text. Solid circles denote support vectors; empty circles denote non-support vectors.

The proof of this theorem is sketched in Appendix B. Figure 2 gives a schematic representation of the quantities that appear in the bound. The bound has a simple geometric intuition: the more distant a non-support vector from the classification hyperplane, the faster its coefficient decays to zero. This is a highly desirable property for large numerical calculations, suggesting that the multiplicative updates could be used to quickly prune away outliers and reduce the size of the quadratic programming problem. Note that while the bound is insensitive to the scale of the inputs, its tightness does depend on their relative locations in the feature space.

## 4 Conclusion

SVMs represent one of the most widely used architectures in machine learning. In this paper, we have derived simple, closed form multiplicative updates for solving the nonnegative quadratic programming problem in SVMs. The $M^3$ updates are straightforward to implement and have a rigorous guarantee of monotonic convergence. It is intriguing that multiplicative updates derived from auxiliary functions appear in so many other areas of machine learning, especially those involving sparse, nonnegative optimizations. Examples include the Baum-Welch algorithm[1] for discrete hidden markov models, generalized iterative scaling[6] and adaBoost[4] for logistic regression, and nonnegative matrix factorization[11, 12] for dimensionality reduction and feature extraction. In these areas, simple multiplicative updates with guarantees of monotonic convergence have emerged over time as preferred methods of optimization. Thus it seems worthwhile to explore their full potential for SVMs.

## Footnotes

[1]The extensions to non-realizable data sets and to hyperplanes that do not pass through the origin are straightforward. They will be treated in a longer paper.

[2] In fact, the multiplicative updates by nature cannot directly set a variable to zero. However, a variable can be clamped to zero whenever its value falls below some threshold (e.g., machine precision) and when a zero value would satisfy the Karush-Kuhn-Tucker conditions.

## References

[1] L. Baum. An inequality and associated maximization technique in statistical estimation of probabilistic functions of Markov processes. *Inequalities*, 3:1–8, 1972.

[2] C. L. Blake and C. J. Merz. UCI repository of machine learning databases, 1998.

[3] C. J. C. Burges. A tutorial on support vector machines for pattern recognition. *Knowledge Discovery and Data Mining*, 2(2):121–167, 1998.

[4] M. Collins, R. Schapire, and Y. Singer. Logistic regression, adaBoost, and Bregman distances. In *Proceedings of the Thirteenth Annual Conference on Computational Learning Theory*, 2000.

[5]  N. Cristianini, C. Campbell, and J. Shawe-Taylor. Multiplicative updatings for support vector machines. In *Proceedings of ESANN'99*, pages 189–194, 1999.

[6]  J. N. Darroch and D. Ratcliff. Generalized iterative scaling for log-linear models. *Annals of Mathematical Statistics*, 43:1470–1480, 1972.

[7]  A. P. Dempster, N. M. Laird, and D. B. Rubin. Maximum likelihood from incomplete data via the EM algorithm. *Journal of the Royal Statistical Society B*, 39:1–37, 1977.

[8]  C. Gentile. A new approximate maximal margin classification algorithm. *Journal of Machine Learning Research*, 2:213–242, 2001.

[9]  R. P. Gorman and T. J. Sejnowski. Analysis of hidden units in a layered network trained to classify sonar targets. *Neural Networks*, 1(1):75–89, 1988.

[10]  J. Kivinen and M. Warmuth. Exponentiated gradient versus gradient descent for linear predictors. *Information and Computation*, 132(1):1–63, 1997.

[11]  D. D. Lee and H. S. Seung. Learning the parts of objects with nonnegative matrix factorization. *Nature*, 401:788–791, 1999.

[12]  D. D. Lee and H. S. Seung. Algorithms for non-negative matrix factorization. In T. K. Leen, T. G. Dietterich, and V. Tresp, editors, *Advances in Neural and Information Processing Systems*, volume 13, Cambridge, MA, 2001. MIT Press.

[13]  O. L. Mangasarian and D. R. Musicant. Lagrangian support vector machines. *Journal of Machine Learning Research*, 1:161–177, 2001.

[14]  O. L. Mangasarian and W. H. Wolberg. Cancer diagnosis via linear programming. *SIAM News*, 23(5):1–18, 1990.

[15]  J. Platt. Fast training of support vector machines using sequential minimal optimization. In B. Schölkopf, C. J. C. Burges, and A. J. Smola, editors, *Advances in Kernel Methods — Support Vector Learning*, pages 185–208, Cambridge, MA, 1999. MIT Press.

[16]  L. K. Saul and D. D. Lee. Multiplicative updates for classification by mixture models. In T. G. Dietterich, S. Becker, and Z. Ghahramani, editors, *Advances in Neural and Information Processing Systems*, volume 14, Cambridge, MA, 2002. MIT Press.

[17]  B. Schölkopf and A. J. Smola. *Learning with Kernels*. MIT Press, Cambridge, MA, 2002.

[18]  V. Vapnik. *Statistical Learning Theory*. Wiley, N.Y., 1998.

## A   Proof of Theorem 1

The proof of monotonic convergence in the objective function $F(\mathbf{v})$, eq. (1), is based on the derivation of an auxiliary function. Similar techniques have been used for many models in statistical learning[1, 4, 6, 7, 12, 16]. An auxiliary function $G(\tilde{\mathbf{v}}, \mathbf{v})$ has the two crucial properties that $F(\tilde{\mathbf{v}}) \leq G(\tilde{\mathbf{v}}, \mathbf{v})$ and $F(\mathbf{v}) = G(\mathbf{v}, \mathbf{v})$ for all nonnegative $\tilde{\mathbf{v}}, \mathbf{v}$. From such an auxiliary function, we can derive the update rule $\mathbf{v}' = \arg\min_{\tilde{\mathbf{v}}} G(\tilde{\mathbf{v}}, \mathbf{v})$ which never increases (and generally decreases) the objective function $F(\mathbf{v})$:

$$F(\mathbf{v}') \leq G(\mathbf{v}', \mathbf{v}) \leq G(\mathbf{v}, \mathbf{v}) = F(\mathbf{v}). \tag{10}$$

By iterating this procedure, we obtain a series of estimates that improve the objective function. For nonnegative quadratic programming, we derive an auxiliary function $G(\tilde{\mathbf{v}}, \mathbf{v})$ by decomposing $F(\mathbf{v})$ in eq. (1) into three terms and then bounding each term separately:

$$F(\mathbf{v}) = \frac{1}{2} \sum_{ij} A_{ij}^{+} v_i v_j - \frac{1}{2} \sum_{ij} A_{ij}^{-} v_i v_j + \sum_i b_i v_i, \tag{11}$$

$$G(\tilde{\mathbf{v}}, \mathbf{v}) = \frac{1}{2} \sum_i \frac{(\mathbf{A}^{+}\mathbf{v})_i}{v_i} \tilde{v}_i^2 - \frac{1}{2} \sum_{ij} A_{ij}^{-} v_i v_j \left(1 + \log \frac{\tilde{v}_i \tilde{v}_j}{v_i v_j}\right) + \sum_i b_i \tilde{v}_i. \tag{12}$$

It can be shown that $F(\tilde{\mathbf{v}}) \leq G(\tilde{\mathbf{v}}, \mathbf{v})$. The minimization of $G(\tilde{\mathbf{v}}, \mathbf{v})$ is performed by setting its derivative to zero, leading to the multiplicative updates in eq. (3). The updates

move each element $v_i$ in the same direction as $-\partial F/\partial v_i$, with fixed points occurring only if $v_i^* = 0$ or $\partial F/\partial v_i = 0$. Since the overall optimization is convex, all minima of $F(\mathbf{v})$ are global minima. The updates converge to the unique global minimum if it exists.

# B   Proof of Theorem 2

The proof of the bound on the asymptotic rate of convergence relies on the repeated use of equalities and inequalities that hold at the fixed point $\alpha^*$. For example, if $\alpha_i^* = 0$ is a non-support vector coefficient, then $(\partial L/\partial \alpha_i)|_{\alpha^*} \geq 0$ implies $(\mathbf{A}^+\alpha^*)_i - (\mathbf{A}^-\alpha^*)_i \geq 1$. As shorthand, let $z_i^+ = (\mathbf{A}^+\alpha^*)_i$ and $z_i^- = (\mathbf{A}^-\alpha^*)_i$. Then we have the following result:

$$\frac{1}{\gamma_i} = \frac{2z_i^+}{1 + \sqrt{1 + 4z_i^+ z_i^-}} \tag{13}$$

$$\geq \frac{2z_i^+}{1 + \sqrt{(z_i^+ - z_i^-)^2 + 4z_i^+ z_i^-}} \tag{14}$$

$$= \frac{2z_i^+}{1 + z_i^+ + z_i^-} = 1 + \frac{z_i^+ - z_i^- - 1}{z_i^+ + z_i^- + 1} \tag{15}$$

$$\geq 1 + \frac{z_i^+ - z_i^- - 1}{2z_i^+}. \tag{16}$$

To prove the theorem, we need to express this result in terms of kernel dot products. We can rewrite the variables in the numerator of eq. (16) as:

$$z_i^+ - z_i^- = \sum_j A_{ij}\alpha_j^* = \sum_j y_i y_j K(\mathbf{x}_i, \mathbf{x}_j)\alpha_j^* = y_i K(\mathbf{x}_i, \mathbf{w}) = |K(\mathbf{x}_i, \mathbf{w})|, \tag{17}$$

where $\mathbf{w} = \sum_j \alpha_j^* \mathbf{x}_j y_j$ is the normal vector to the maximum margin hyperplane. Likewise, we can obtain a bound on the denominator of eq. (16) by:

$$z_i^+ = \sum_j A_{ij}^+ \alpha_j^* \tag{18}$$

$$\leq \max_k A_{ik}^+ \sum_j \alpha_j^* \tag{19}$$

$$\leq \max_k |K(\mathbf{x}_i, \mathbf{x}_k)| \sum_j \alpha_j^* \tag{20}$$

$$\leq \sqrt{K(\mathbf{x}_i, \mathbf{x}_i)} \max_k \sqrt{K(\mathbf{x}_k, \mathbf{x}_k)} \sum_j \alpha_j^* \tag{21}$$

$$= \sqrt{K(\mathbf{x}_i, \mathbf{x}_i)} \max_k \sqrt{K(\mathbf{x}_k, \mathbf{x}_k)} K(\mathbf{w}, \mathbf{w}). \tag{22}$$

Eq. (21) is an application of the Cauchy-Schwartz inequality for kernels, while eq. (22) exploits the observation that:

$$K(\mathbf{w}, \mathbf{w}) = \sum_{jk} A_{jk}\alpha_j^* \alpha_k^* = \sum_j \alpha_j^* \sum_k A_{jk}\alpha_k^* = \sum_j \alpha_j^*. \tag{23}$$

The last step in eq. (23) is obtained by recognizing that $\alpha_j^*$ is nonzero only for the coefficients of support vectors, and that in this case the optimality condition $(\partial L/\partial \alpha_j)|_{\alpha^*} = 0$ implies $\sum_k A_{jk}\alpha_k^* = 1$. Finally, substituting eqs. (17) and (22) into eq. (16) gives:

$$\frac{1}{\gamma_i} \geq 1 + \frac{|K(\mathbf{x}_i, \mathbf{w})| - 1}{2\sqrt{K(\mathbf{x}_i, \mathbf{x}_i)} \max_k \sqrt{K(\mathbf{x}_k, \mathbf{x}_k)} K(\mathbf{w}, \mathbf{w})}. \tag{24}$$

This reduces in a straightforward way to the claim of the theorem.